# A Neurocomputer Board Based on the ANNA Neural Network Chip

Eduard Säckinger, Bernhard E. Boser, and Lawrence D. Jackel
AT&T Bell Laboratories
Crawfords Corner Road, Holmdel, NJ 07733

## Abstract

A board is described that contains the ANNA neural-network chip, and a DSP32C digital signal processor. The ANNA (Analog Neural Network Arithmetic unit) chip performs mixed analog/digital processing. The combination of ANNA with the DSP allows high-speed, end-to-end execution of numerous signal-processing applications, including the preprocessing, the neural-net calculations, and the postprocessing steps. The ANNA board evaluates neural networks 10 to 100 times faster than the DSP alone. The board is suitable for implementing large (million connections) networks with sparse weight matrices. Three applications have been implemented on the board: a convolver network for slant detection of text blocks, a handwritten digit recognizer, and a neural network for recognition-based segmentation.

## 1  INTRODUCTION

Many researchers have built neural-network chips, but few chips have been installed in board-level systems, even though this next level of integration provides insights and advantages that can't be attained on a chip testing station. Building a board demonstrates whether or not the chip can be effectively integrated into the larger systems required for real applications. A board also exposes bottlenecks in the system data paths. Most importantly, a working board moves the neural-network chip from the realm of a research exercise, to that of a practical system, readily available to users whose primary interest is actual applications. An additional bonus of carrying the integration to the board level is that the chip designer can gain the user feedback that will assist in designing new chips with greater utility.

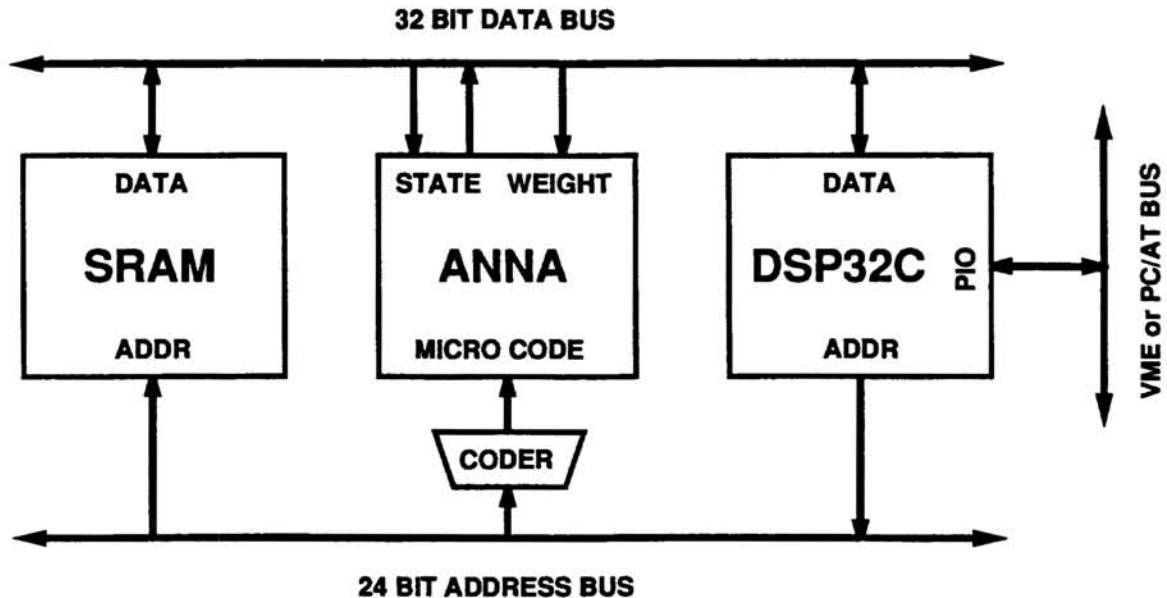

Figure 1: Block Diagram of the ANNA Board

## 2    ARCHITECTURE

The neurocomputer board contains a special purpose chip called ANNA (Boser et al., 1991), for the parallel evaluation of neuron functions (a squashing function applied to a weighted sum) and a general purpose digital signal processor, DSP32C. The board also contains interface and clock synchronization logic as well as 1 MByte of static memory, SRAM (see Fig. 1). Two version of this board with two different bus interfaces have been built: a double height VME board (see Fig. 2) and a PC/AT board (see Fig. 3).

The ANNA neural network chip is an ALU (Arithmetic and Logic Unit) specialized for neural network functions. It contains a 12-bit wide state-data input, a 12-bit wide state-data output, a 12-bit wide weight-data input, and a 37-bit micro-instruction input. The instructions that can be executed by the chip are the following (parameters are not shown):

**RFSH** Write weight values from the weight-data input into the dynamic on-chip weight storage.

**SHIFT** Shift on-chip barrel shifter to the left and load up to four new state values from state-data input into the right end of the shifter.

**STORE** Transfer state vector from the shifter into the on-chip state storage and/or into the state-data latches of the arithmetic unit.

**CALC** Calculate eight dot-products between on-chip weight vectors and the contents of the above mentioned data latches; subsequently evaluate the squashing function.

**OUT** Transfer the results of the calculation to the state-data output.

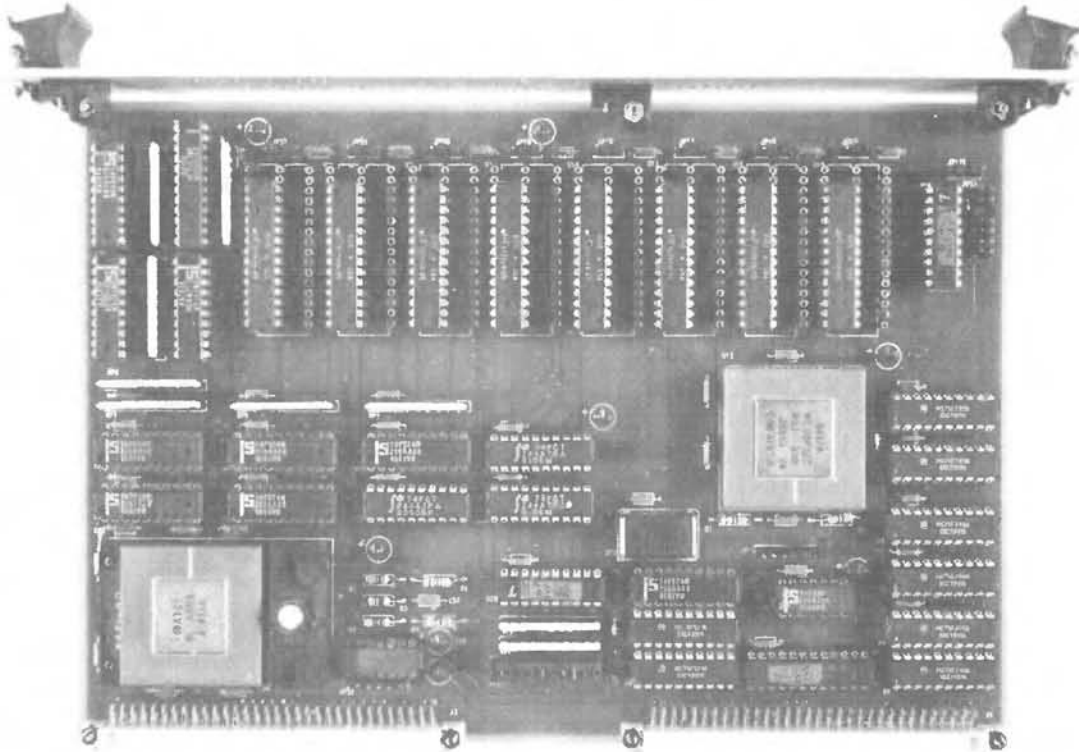

Figure 2: ANNA Board with VME Bus Interface

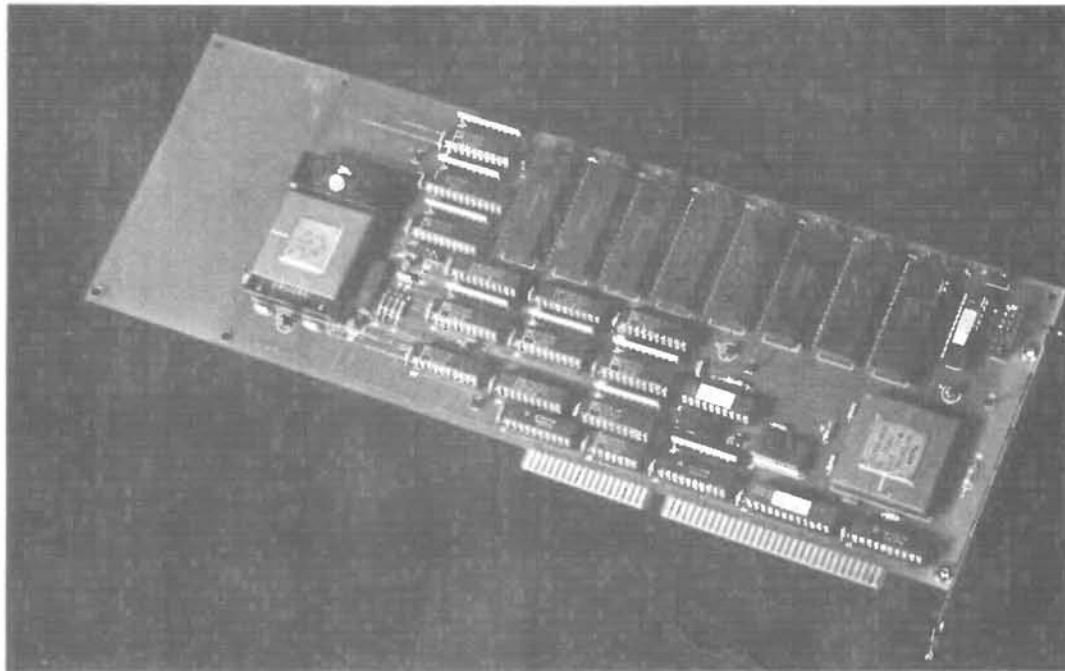

Figure 3: ANNA Board with PC/AT Bus Interface

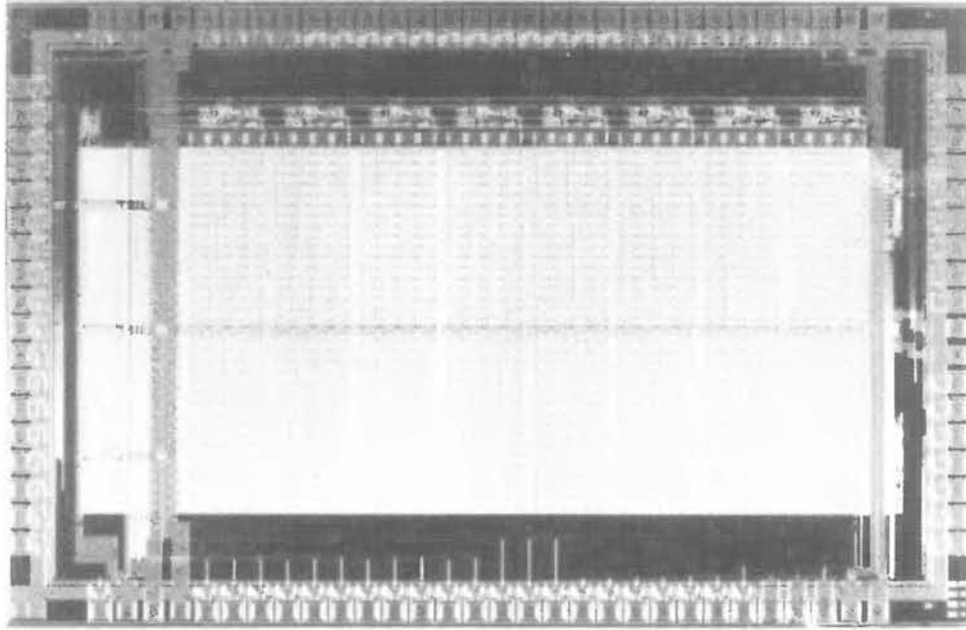

Figure 4: Photo Micrograph of the ANNA Chip

Some of the instructions (like SHIFT and CALC) can be executed in parallel. The barrel shifter at the input as well as the on-chip state storage make the ANNA chip very effective for evaluating locally-connected, weight-sharing networks such as feature extraction and time-delay neural networks (TDNN).

The ANNA neural network chip, implemented in a $0.9\,\mu$m CMOS technology, contains 180,000 transistors on a $4.5 \times 7\,\text{mm}^2$ die (see Fig. 4). The chip implements 4,096 *physical* synapses which can be time multiplexed in order to realize networks with many more than 4,096 connections. The resolution of the synaptic weights is 6 bits and that of the states (input/output of the neurons) is 3 bits. Additionally, a 4-bit scaling factor can be programmed for each neuron to extend the dynamic range of the weights. The weight values are stored as charge packets on capacitors and are periodically refreshed by two on chip 6-bit D/A converter. The synapses are realized by multiplying 3-bit D/A converters (analog weight times digital state). The analog results of this multiplication are added by means of current summing and then converted back to digital by a saturating 3-bit A/D converter. Although the chip uses analog computing internally, all input/output is digital. This combines the advantages of the high synaptic density, the high speed, and the low power of analog with the ease of interfacing to a digital system like a digital signal processor (DSP).

The 32-bit floating-point digital signal processor (DSP32C) on the same board runs at 40 MHz without wait states (100 ns per instruction) and is connected to 1 MByte of static RAM. The DSP has several functions: (1) It generates the micro instructions for the ANNA chip. (2) It is responsible for accessing the pixel, feature, and weight data from the memory and then storing the results of the chip in the memory. (3) If the precision of the ANNA chip is not sufficient the DSP can do the calculations with 32-bit floating-point precision. (4) Learning algorithms can be run

on the DSP. (5) The DSP is useful as a pre- and post-processor for neural networks. In this way a whole task can be carried out on the board without exchanging intermediate results with the host.

As shown by Fig. 1 ANNA instructions are supplied over the DSP address bus, while state and weight data is transferred over the data bus. This arrangement makes it possible to supply or store ANNA data *and* execute a micro instruction simultaneously, i.e., using only one DSP instruction. The ANNA clock is automatically generated whenever the DSP issues a micro instruction to the ANNA chip.

## 3   PERFORMANCE

Using a DSP for supplying micro instructions as well as accessing the data from the memory makes the board very flexible and fairly simple. Both data and instruction flow to and from the ANNA chip are under software control and can be programmed using the C or DSP32C assembly language.

Because of DSP32C features such as one-instruction 32-bit memory-to-memory transfer with auto increment and overhead free looping, ANNA instruction sequences can be generated at a rate of approximately 5 MIPS. A similar rate of 5 MByte/s is achieved for reading and writing ANNA data from and to the memory.

The speed of the board depends on the application and how well it makes use of the chip's parallelism and ranges between 30 MC/s and 400 MC/s. For concrete examples see the section on Applications. Compared to the DSP32C which performs at about 3 MC/s (for sparsely connected networks) the board with the ANNA chip is 10 to 100 times faster.

The speed of the board is not limited by the ANNA chip but by the above mentioned data rates. The use of a dedicated hardware sequencer will improve the speed by up to ten times. The board can thus be used for prototyping an application, before building more specialized hardware.

## 4   SOFTWARE

To make the board easily usable we implemented a LISP interpreter on the host computer (a SUN workstation) which allows us to make remote procedure calls (RPC) to the ANNA board. After starting the LISP interpreter on the host it will download the DSP object code to the board and start the main program on the DSP. Then, the DSP will transfer the addresses of all procedures that are available to the user to the LISP interpreter. From then on, all these procedures can be called as LISP functions of the form (==> anna *procedure parameter(s)*) from the host. Parameters and return value are handled automatically by the LISP interpreter.

Three ways of using the ANNA board are described. The first two methods do not require DSP programming; everything is controlled from the LISP interpreter. The third method requires DSP programming and results in maximum speed for any application.

1. The simplest way to use the board together with this LISP interpreter is to call existing library functions on the board. For example a neural network for recognizing handwritten digits can be called as follows:

```
(==> anna down-weight weight-matrix)
(setq class (==> anna down-rec-up digit-pattern))
```

The first LISP function activates the **down-weight** function on the ANNA board that transfers the LISP matrix, *weight-matrix*, to the board. This function defines all the weights of the network and has to be called only once. The second LISP function calls the **down-rec-up** function which takes the *digit-pattern* (pixel image) as an input, downloads this pattern, runs the recognizer, and uploads the *class* number (0 ...9).

This method requires no knowledge of the ANNA or DSP instruction set. The library functions are fast since they have been optimized by the implementer. At the moment library functions for nonlinear convolution, character recognition, and testing are available.

2. If a function which is not part of the library has to be implemented, an ANNA program must be written. A collection of LISP functions (ANNANAS), support the translation of symbolic ANNA program into micro code. The micro code is then run on the ANNA chip by means of a software sequencer implemented on the DSP. Assembling and running a simple ANNA program using ANNANAS looks like this:

```
(anna-repeat 16)      ; REPEAT 16         start of loop
(anna-shift 4 0)      ;   SHIFT 4,R0;     ANNA shift instruction
(anna-store 0 'a 2)   ;   STORE R0,A.L2;  ANNA store instruction
(anna-endrep)         ; ENDREP            end of loop
(anna-stop)           ; STOP              end of program

(anna-run 0)          ;                   start sequencer
```

In this way, all the features of the ANNA chip and board can be used without DSP programming. This mode is also helpful for testing and debugging ANNA programs. Beside the assembler, ANNANAS also provides several monitoring and debugging tools.

3. If maximum speed is imperative, an application specific sequencer has to be written (as opposed to the slower generic sequencer described above). To do this a DSP assembler and C compiler are required. A toolbox of assembly macros and C functions help implementing this sequencer. Besides the sequencer, pre- and post-processing software can also be implemented on the fast DSP hardware. After successfully testing the program it can be added to the library as a new function.

## 5   APPLICATIONS

### 5.1   CONVOLVER NETWORK

In this application the ANNA chip is configured for 16 neurons with 256 synapses each. First, each of these neurons connect to the upper left 16 × 16 field of a

Table 1: Performance of the Recognizer.

| IMPLEMENTATION | ERROR RATE | REJECT RATE FOR 1% ERROR |
|---|---|---|
| Full Precision | 4.9% | 9.1% |
| ANNA/DSP | 5.3 ± 0.2% | 13.5 ± 0.8% |
| ANNA/DSP/Retraining | 4.9 ± 0.2% | 11.5 ± 0.8% |

large image. The 16 neurons are then scanned horizontally and vertically over the whole image. The nve.

## 5.3 RECOGNITION BASED SEGMENTATION

Before individual digits can be passed to a recognizer as described in the previous section, they typically have to be isolated (segmented) from a string of characters (e.g. a ZIP code). When characters overlap, segmentation is a difficult problem and simple algorithms which look for connected components or histograms fail.

A promising solution to this problem is to combine recognition and segmentation (Keeler et al., 1992, Matan et al., 1992). For instance recognizers like the one described above can be replicated horizontally and vertically over the region of interest. This will guarantee, that there is a recognizer centered over each character. It is crucial, however, to train the recognizer such that it rejects partial characters. Such a replicated version of the recognizer (at 31 times 6 locations) with approximately 2 million connections has been implemented on the ANNA board and was used to segment ZIP codes.

## 6  CONCLUSION

A board with a neural-network chip and a digital signal processor (DSP) has been built. Large pattern recognition applications have been implemented on the board giving a speed advantage of 10 to 100 over the DSP alone.

### Acknowledgements

The authors would like to thank Steve Deiss for his excellent job in building the boards and Yann LeCun and Jane Bromley for their help with the digit recognizer.

### References

Bernhard Boser, Eduard Säckinger, Jane Bromley, Yann LeCun, and Lawrence D. Jackel. An analog neural network processor with programmable network topology. *IEEE J. Solid-State Circuits*, 26(12):2017–2025, December 1991.

Yann Le Cun, Bernhard Boser, John S. Denker, Donnie Henderson, Richard E. Howard, Wayne Hubbard, and Lawrence D. Jackel. Handwritten digit recognition with a back-propagation network. In David S. Touretzky, editor, *Neural Information Processing Systems*, volume 2, pages 396–404. Morgan Kaufmann Publishers, San Mateo, CA, 1990.

Eduard Säckinger, Bernhard Boser, Jane Bromley, Yann LeCun, and Lawrence D. Jackel. Application of the ANNA neural network chip to high-speed character recognition. *IEEE Trans. Neural Networks*, 3(2), March 1992.

J. D. Keeler and D. E. Rumelhart. Self-organizing segmentation and recognition neural network. In J. M. Moody, S. J. Hanson, and R. P. Lippman, editors, *Neural Information Processing Systems*, volume 4. Morgan Kaufmann Publishers, San Mateo, CA, 1992.

Ofer Matan, Christopher J. C. Burges, Yann LeCun, and John S. Denker. Multi-digit recognition using a space delay neural network. In J. M. Moody, S. J. Hanson, and R. P. Lippman, editors, *Neural Information Processing Systems*, volume 4. Morgan Kaufmann Publishers, San Mateo, CA, 1992.